# Bounded invariance and the formation of place fields

**Reto Wyss and Paul F.M.J. Verschure**
Institute of Neuroinformatics
University/ETH Zürich
Zürich, Switzerland
`rwyss,pfmjv@ini.phys.ethz.ch`

## Abstract

One current explanation of the view independent representation of space by the place-cells of the hippocampus is that they arise out of the summation of view dependent Gaussians. This proposal assumes that visual representations show bounded invariance. Here we investigate whether a recently proposed visual encoding scheme called the temporal population code can provide such representations. Our analysis is based on the behavior of a simulated robot in a virtual environment containing specific visual cues. Our results show that the temporal population code provides a representational substrate that can naturally account for the formation of place fields.

## 1 Introduction

Pyramidal cells in the CA3 and CA1 regions of the rat hippocampus have shown to be selectively active depending on the animal's position within an environment[1]. The ensemble of locations where such a cell fires – the *place field* – can be determined by a combination of different environmental and internal cues[2], where vision has been shown to be of particular importance[3]. This raises the question, how egocentric visual representations of visual cues can give rise to an allocentric representation of space. Recently it has been proposed that a place field is formed by the summation of Gaussian tuning curves, each oriented perpendicular to a wall of the environment and peaked at a fixed distance from it[4, 5, 6]. While this proposal tries to explain the actual transformation from one coordinate system to another, it does not account for the problem how appropriate egocentric representations of the environment are formed. Thus, it is unclear, how the information about a rat's distance to different walls becomes available, and in particular how this proposal would generalize to other environments where more advanced visual skills, such as cue identification, are required.

For an agent moving in an environment, visual percepts of objects/cues undergo a combination of transformations comprising zooming and rotation in depth. Thus, the question arises, how to construct a visual detector, which has a Gaussian like tuning with regard to the positions within the environment from which snapshots

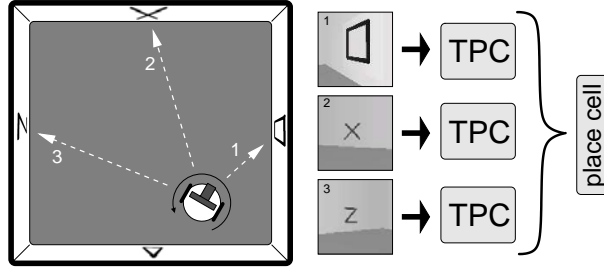

Figure 1: Place cells from multiple snapshots. The robot is placed in a virtual square environment with four patterns on the walls, i.e. a square, a triangle, a Z and a X. The robot scans the environment for salient stimuli by rotating on place. A saliency detector triggers the acquisition of visual snapshots which are subsequently transformed into TPCs. A place cell is defined through its associated TPC templates.

of a visual cue are taken. The internal representation of a stimulus, upon which such a detector is based, should be tolerant to certain degrees of visual deformations without loosing specificity or, in other words, show a bounded invariance. In this study we show that a recently proposed cortical model of visual pattern encoding, the temporal population code (TPC), directly supports this notion of bounded invariance[7]. The TPC is based on the notion that a cortical network can be seen to transform a spatial pattern into a purely temporal code.

Here, we investigate to what extent the bounded invariance provided by the TPC can be exploited for the formation of place fields. We address this question in the context of a virtual robot behaving in an environment containing several visual cues. Our results show, that the combination of a simple saliency mechanism with the TPC naturally gives rise to allocentric representations of space, similar to the place fields observed in the hippocampus.

## 2 Methods

### 2.1 The experimental setup

Experiments are performed using a simulated version of the real-world robot *Khepera* (K-team, Lausanne, Switzerland) programmed in C++ using OpenGL. The robot has a circular body with two wheels attached to its side each controlled by an individual motor. The visual input is provided by a camera with a viewing angle of $60°$ mounted on top of the robot. The neural networks are simulated on a Linux computer using a neural network simulator programmed in C++.

The robot is placed in square arena (fig. 1, left),and in the following, all lengths will be given in units of the side lengths of the square environment.

### 2.2 The temporal population code

Visual information is transformed into a TPC by a network of laterally coupled cortical columns, each selective to one of four orientations $\psi \in \{0°, 45°, 90°, 135°\}$ and one of three spatial frequencies $\nu \in \{high, medium, low\}$[7]. The outputs of the network are twelve vectors $\mathbf{A}_{\psi,\nu}$ each reflecting the average population activity recorded over 100 time-steps for each type of cortical column. These vectors are

reduced to three vectors $\mathbf{A}_\nu$ by concatenating the four orientations. This set of vectors form the TPC which represents a single snapshot of a visual scene.

The similarity $S(s_1, s_2)$ between two snapshots $s_1$ and $s_2$ is defined as the average correlation $\rho$ between the corresponding vectors, i.e.

$$S(s_1, s_2) \quad = \quad \left\langle Z\Big(\rho(\mathbf{A}_\nu^{s_1}, \mathbf{A}_\nu^{s_2})\Big)\right\rangle_{\forall \nu} \qquad (1)$$

where $Z$ is the Fisher Z-Transform given by $Z(\rho) = 1/2\ln((1+\rho)/(1-\rho))$, which transforms a typically skewed distribution of correlation coefficients $\rho$ into an approximately normal distribution of coefficients. Thus, $Z(\rho)$ becomes a measure on a proportional scale such that mean values are well defined.

## 2.3  Place cells from multiple snapshots

In this study, the response properties of a place cell are given by the similarity between incoming snapshots of the environment and template snapshots associated to the place cell when it was constructed. Thus, for both, the acquisition of place cells as well as their exploitation, the system needs to be provided with snapshots of its environment that contain visual features. For this purpose, the robot is equipped with a simple visual saliency detector $s(t)$ that selects scenes with high central contrast:

$$s(t) = \frac{\sum e^{-\mathbf{y}^2} c(\mathbf{y}, t)^2}{\sum c(\mathbf{y}, t)^2}$$

where $c(\mathbf{y}, t)$ denotes the contrast at location $\mathbf{y} \in [-1, +1]^2$ in the image at time $t$. At each point in time where $s(t) > \theta_{saliency}$, a new snapshot is acquired with a probability of 0.1. A place cell $k$ is defined by $n$ snapshots called templates $t_i^k$ with $i = 1 \ldots n$.

Whenever the robot tries to localize itself, it scans the environment by rotating in place and taking snapshots of visually salient scenes (fig. 1). The similarity $S$ between each incoming snapshot $s_j$ with $j = 1 \ldots m$ and every template $t_i^k$ is determined using eq. 1. The activation $a_k$ of place cell $k$ for a series of $m$ snapshots $s_j$ is then given by a sigmoidal function

$$a_k(i_k) = \Big(1 + \exp\big(-\beta(i_k - \theta)\big)\Big)^{-1} \quad \text{where} \quad i_k = \left\langle \max_i \Big(S(t_i^k, s_j)\Big)\right\rangle_j. \qquad (2)$$

$i_k$ represents the input to the place cell which is computed by determining the maximal similarity of each snapshot to any template of the place cell and subsequent averaging, i.e. $\langle \cdot \rangle_j$ corresponds to the average over all snapshots $j$.

## 2.4  Position reconstruction

There are many different approaches to the problem of position reconstruction or decoding from place cell activity[8]. A basis function method uses a linear combination of basis functions $\phi_k(\mathbf{x})$ with the coefficients proportional to the activity of the place cells $a_k$. Here we use a direct basis approach, i.e. the basis function $\phi_k(\mathbf{x})$ directly corresponds to the average activation $a_k$ of place cell $k$ at position $\mathbf{x}$ within the environment. The reconstructed position $\hat{\mathbf{x}}$ is then given by

$$\hat{\mathbf{x}} = \underset{\mathbf{x}}{\operatorname{argmax}} \sum_k a_k \phi_k(\mathbf{x})$$

The reconstruction error is given by the distance between the reconstructed and true position averaged over all positions within the environment.

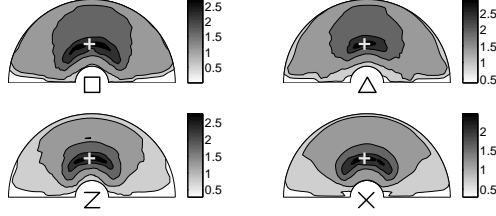

Figure 2: Similarity surfaces for the four different cues. Similarity between a reference snapshot of the different cues taken at the position marked by the white cross and all the other positions surrounding the reference location.

## 2.5 Place field shape and size

In order to investigate the shape of a place field $\phi(\mathbf{x})$, and in particular to determine its degree of asymmetry and its size, we computed the two-dimensional normalized inertial tensor $\mathbf{I}$ given by

$$I_{ij} = \frac{\sum_{\mathbf{r}} \phi(\mathbf{r})\left(\delta_{ij}\mathbf{r}^2 - r_i r_j\right)}{\sum_{\mathbf{r}} \phi(\mathbf{r})}$$

with $\mathbf{r} = \{r_1, r_2\} = \mathbf{x} - \hat{\mathbf{x}}$ where $\hat{\mathbf{x}} = \sum \mathbf{x}\phi(\mathbf{x})/\sum \phi(\mathbf{x})$ corresponds to the "center of gravity" and $\delta_{ij}$ is the Kronecker delta. $\mathbf{I}$ is symmetric and can therefore be diagonalized, i.e. $\mathbf{I} = \mathbf{V^T D V}$, such that $\mathbf{V}$ is an orthonormal transformation matrix and $D_{ii} > 0$ for $i = 1, 2$. A measure of the half-width of the place field along its two principal axes is then $d_i = \sqrt{2D_{ii}}$ such that a measure of asymmetry is given by

$$0 \le \left| \frac{d_1 - d_2}{d_1 + d_2} \right| \le 1$$

This measure becomes zero for symmetric place fields while approaching one for asymmetric ones. In addition, we can estimate the size of the place field by approximating its shape by an ellipse, i.e. $\pi d_1 d_2$.

# 3 Results

## 3.1 Bounded invariance

Initially, we investigate the topological properties of the temporal population coding space. Depending on the position within an environment, visual stimuli undergo a geometric transformation which is a combination of scaling and rotation in depth. Fig. 2 shows the similarity to a reference snapshot taken at the location of the white cross for the four different cues. Although the precise shape of the similarity surface differs, the similarity decreases smoothly and monotonically for increasing distances to the reference point for all stimuli.

The similarity surface for different locations of the reference point is shown in fig. 3 for the Z cue. Although the Z cue has no vertical mirror symmetry, the similarity surfaces are nearly symmetric with respect to the vertical center line. Thus, using a single cue, localization is only possible modulo a mirror along the vertical center. The implications of this will be discussed later. Concerning different distances of the reference point to the stimulus, fig. 3 (along the columns) shows that the specificity of the similarity measure is large for small distances while the tuning becomes

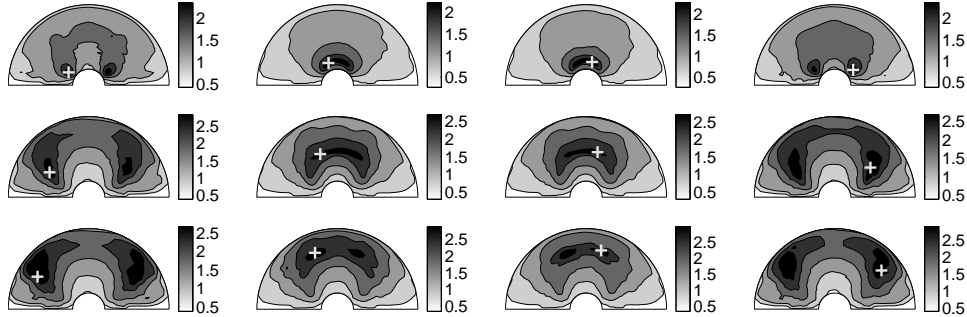

Figure 3: Similarity surface of Z cue for different reference points. The distance/angle of the reference point to the cue is kept constant along the rows/columns respectively.

broader for large distances. This is a natural consequence of the perspective projection which implies that the changes in visual perception due to different viewing positions are inversely proportional to the viewing distance.

## 3.2 Place cells from multiple snapshots

The response of a place cell is determined by eq. 2 based on four associated snapshots/templates taken at the same location within the environment. The templates for each place cell are chosen by the saliency detector and therefore there is no explicit control over the actual snapshots defining a place cell, i.e. some place cells are defined based on two or more templates of the same cue. Furthermore, the stochastic nature of the saliency detector does not allow for any control over the precise position of the stimulus within the visual field. This is, where the intrinsic translation invariance of the temporal population code plays an important role, i.e. the precise position of the stimulus within the visual field at the time of the snapshot has no effect on the resulting encoding as long as the whole stimulus is visible.

Fig. 4 shows examples of the receptive fields (subsequently also called place fields) of such place cells acquired at the nodes of a regular $5 \times 5$ lattice within the environment. Most of the place fields have a Gaussian-like tuning which is compatible with single cell recordings from pyramidal cells in CA3 and CA1[2], i.e. the place cells maximally respond close to their associated positions and degrade smoothly and monotonically for increasing distances. Some place cells have multiple subfields in that they respond to different locations in the environment with a similar amplitude.

## 3.3 Position reconstruction

Subsequently, we determine the accuracy up to which the robot can be localized within the environment. Therefore we use the direct basis approach for position reconstruction as described in the *Methods*. As basis functions we take the normalized response profiles of place cells constructed from four templates taken at the nodes of a regular lattice covering the environment. Fig. 5a shows the reconstruction error averaged over the environment as a function of the number of place cells as well as the number of snapshots taken at each location. The reconstruction error decreases monotonically both for an increasing number of place cells as well as an increasing

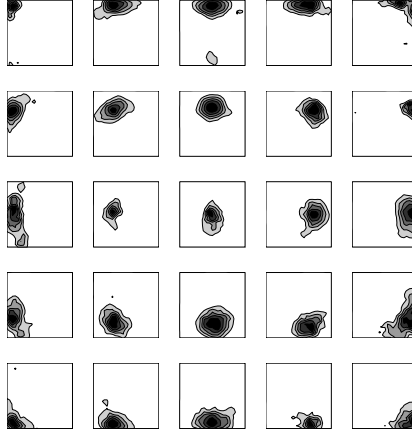

Figure 4: Place fields of 5 × 5 place cells. The small squares show the average response of 5 × 5 different place cells for all the positions of the robot within the environment. Darker regions correspond to stronger responses. The relative location of each square within the figure corresponds to the associated location of the place cell within the environment. All place fields are scaled to a common maximal response.

number of snapshots. An asymptotic reconstruction error is approached very fast, i.e. for more then 25 place cells and more then two snapshots per location. Thus, for a behaving organism exploring an unknown environment, this implies that a relatively sparse exploration strategy suffices to create a complete representation of the new environment.

Above we have seen that localization with a single snapshot is only possible modulo a mirror along the axis where the cue is located. The systematic reconstruction error introduced by this short-coming can be determined analytically and is $\approx 0.13$ in units of the side-length of the square environment. For an increasing number of snapshots, the probability that all snapshots are from the same pair of opposite cues, decreases exponentially fast and we therefore also expect the systematic error to vanish. Considering 100 place cells, the difference in reconstruction error between 1 and 10 snapshots amounts to $0.147 \pm 0.008$ (mean $\pm$ SD) which is close to the predicted systematic error due to the effect discussed above. Thus, an increasing number of snapshots primarily helps to resolve ambiguities due to the symmetry properties of the TPC.

### 3.4 Place field shape

Fig. 5b-c shows scatter plots of both, place field asymmetry and size versus the distance of the place field's associated location from the center of the environment. There is a tendency that off-center place cells have more asymmetric place fields than cells closer to the center (r=0.32) which is in accordance with experimental results[5]. Regarding place field size, there is no direct relation to the associated position of place field (r=0.08) apart from the fact that the variance is maximal for intermediate distances from the center. It must be noted, however, that the size of the place field critically depends on the choice of the threshold $\theta$ in eq. 2. Indeed different relations between place field size and location can be achieved by assuming non homogeneous thresholds, which for example might be determined for

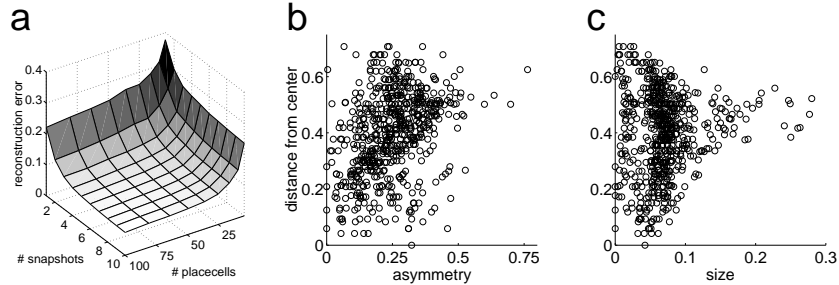

Figure 5: **(a)** Position reconstruction error. The average error in position reconstruction as a function of the number of snapshots and the number of place cells considered. **(b-c)** Scatter plots of the place field asymmetry/size versus the distance of the place fields associated location to the center of the environment. The correlation coefficients are r=0.32/0.08 respectively.

each place cell individually based on its range of inputs. The measure for place field asymmetry, in contrast, has shown to be more stable in this respect (data not shown).

## 4    Discussion

We have shown that the bounded invariance properties of visual stimuli encoded in a TPC are well suited for the formation of place fields. More specifically, the topology preservation of similarity amongst different viewing angles and distances allows a direct translation of the visual similarity between two views to their relative location within an environment. Therefore, only a small number of place cells are required for position reconstruction. Regarding the shape of the place fields, only weak correlations between its asymmetry and its distance to the center of the environment have been found.

As opposed to the present approach, experimental results suggest that place field formation in the hippocampus relies on multiple sensory modalities and not only vision. Although it was shown that vision may play an important role[3], proprioceptive stimuli, for example, can become important in situations where either visual information is not available such as in the dark or in the presence of visual singularities, where two different locations elicit the same visual sensation[9]. A type of information strongly related to proprioceptive stimuli, is the causal structure of behavior which imposes continuous movement in both space and time, i.e. the information about the last location can be of great importance for estimating the current location[10]. Indeed, a recent study has shown that position reconstruction error greatly reduces, if this additional constraint is taken into account[8]. In the present approach we analyzed the properties of place cells in the absence of a behavioral paradigm. Thus, it is not meaningful to integrate information over different locations. We expect, however, that for a continuously behaving robot this type of information would be particularly useful to resolve the ambiguities introduced by the mirror invariance in the case of a single visual snapshot.

As opposed to the large field of view of rats ($\approx 320°$[11]) the robot used in this study has a very restricted field of view. This has direct implications on the robot's behavior. The advantage of only considering a 60° field of view is, however, that the amount of information contributed by single cues can be investigated. We

have shown, that a single view allows for localization modulo a mirror along the orientation of the corresponding stimulus. This ambiguity can be resolved taking additional snapshots into account. In this context, maximal additional information can be gained if a new snapshot is taken along a direction orthogonal to the first snapshot which is also more efficient from a behavioral point of view than using stimuli from opposite directions.

The acquisition of place cells was supervised, in that their associated locations are assumed to correspond to the nodes of a regular lattice spanning the environment. While this allows for a controlled statistical analysis of the place cell properties, it is not very likely that an autonomously behaving agent can acquire place cells in such a regular fashion. Rather, place cells have to be acquired incrementally based on purely local information. Information about the number of place cells responding or the maximal response of any place cell for a particular location is locally available to the agent, and can therefore be used to selectively trigger the acquisition of new place cells. In general, the representation will most likely also reflect further behavioral requirements in that important locations where decisions need to be taken, will be represented by a high density of place cells.

## Acknowledgments

This work was supported by the European Community/Bundesamt für Bildung und Wissenschaft Grant IST-2001-33066 (to P.V.). The authors thank Peter König for valuable discussions and contributions to this study.

## References

[1] J. O'Keefe and J. Dostrovsky. The hippocampus as a spatial map: preliminary evidence from unit activity in the freely moving rat. *Brain Res*, 34:171–5, 1971.

[2] J. O'Keefe and L. Nadel. *The hippocampus as a cognitive map.* Clarendon Press, Oxford, 1987.

[3] J. Knierim, H. Kudrimoti, and B. McNaughton. Place cells, head direction cells, and the learning of landmark stability. *J. Neursci.*, 15:1648–59, 1995.

[4] J. O'Keefe and N. Burgess. Geometric determinants of the place fields of hippocampal neurons. *Nature*, 381(6581):425–8, 1996.

[5] J. O'Keefe, N. Burgess, J.G. Donnett, K.J. Jeffrey, and E.A. Maguire. Place cells, navigational accuracy, and the human hippocampus. *Philos Trans R Soc Lond B Biol Sci.*, 353(1373):1333–40, 1998.

[6] N. Burgess, J.G. Donnett, H.J. Jeffrey, and J. O'Keefe. Robotic and neuronal simulation of the hippocampus and rat navigation. *Philos Trans R Soc Lond B Biol Sci.*, 352(1360):1535–43, 1997.

[7] R. Wyss, P. König, and P.F.M.J. Verschure. Invariant representations of visual patterns in a temporal population code. *Proc. Natl. Acad. Sci. USA*, 100(1):324–9, 2003.

[8] K. Zhang, I. Ginzburg, B.L. McNaughton, and T.J. Sejnowski. Interpreting neuronal population activity by reconstruction: Unified framework with application in hippocampal place cells. *J Neurophysiol.*, 79(2):1017–44, 1998.

[9] A. Arleo and W. Gerstner. Spatial cognition and neuro-mimetic navigation: a model of hippocampal place cell activity. *Biol Cybern.*, 83(3):287–99, 2000.

[10] G. Quirk, R. Muller, and R. Kubie. The firing of hippocampal place cells in the dark depends on the rat's recent experience. *J. Neursci.*, 10:2008–17, 1995.

[11] A. Hughes. A schematic eye for the rat. *Visual Res.*, 19:569–88, 1977.
